# Modeling Natural Sounds with Modulation Cascade Processes

**Richard E. Turner and Maneesh Sahani**
Gatsby Computational Neuroscience Unit
17 Alexandra House, Queen Square, London, WC1N 3AR, London

## Abstract

Natural sounds are structured on many time-scales. A typical segment of speech, for example, contains features that span four orders of magnitude: Sentences ($\sim 1$ s); phonemes ($\sim 10^{-1}$ s); glottal pulses ($\sim 10^{-2}$ s); and formants ($\lesssim 10^{-3}$ s). The auditory system uses information from each of these time-scales to solve complicated tasks such as auditory scene analysis [1]. One route toward understanding how auditory processing accomplishes this analysis is to build neuroscience-inspired algorithms which solve similar tasks and to compare the properties of these algorithms with properties of auditory processing. There is however a discord: Current machine-audition algorithms largely concentrate on the shorter time-scale structures in sounds, and the longer structures are ignored. The reason for this is two-fold. Firstly, it is a difficult technical problem to construct an algorithm that utilises both sorts of information. Secondly, it is computationally demanding to simultaneously process data both at high resolution (to extract short temporal information) and for long duration (to extract long temporal information). The contribution of this work is to develop a new statistical model for natural sounds that captures structure across a wide range of time-scales, and to provide efficient learning and inference algorithms. We demonstrate the success of this approach on a missing data task.

## 1 Introduction

Computational models for sensory processing are still in their infancy, but one promising approach has been to compare aspects of sensory processing with aspects of machine-learning algorithms crafted to solve the same putative task. A particularly fruitful approach in this vein uses the generative modeling framework to derive these learning algorithms. For example, Independent Component Analysis (ICA) and Sparse Coding (SC), Slow Feature Analysis (SFA), and Gaussian Scale Mixture Models (GSMMs) are examples of algorithms corresponding to generative models that show similarities with visual processing [3]. In contrast, there has been much less success in the auditory domain, and this is due in part to the paucity of flexible models with an explicit temporal dimension (although see [2]). The purpose of this paper is to address this imbalance.

This paper has three parts. In the first we review models for the short-time structure of sound and argue that a probabilistic time-frequency model has several distinct benefits over traditional time-frequency representations for auditory modeling. In the second we review a model for the long-time structure in sounds, called probabilistic amplitude demodulation. In the third section these two models are combined with the notion of auditory features to produce a full generative model for sounds called the Modulation Cascade Process (MCP). We then show how to carry out learning and inference in such a complex hierarchical model, and provide results on speech for complete and missing data tasks.

## 2 Probabilistic Time-Frequency Representations

Most representations of sound focus on the short temporal structures. Short segments ($< 10^{-1}$ s) are frequently periodic and can often be efficiently represented in a Fourier basis as the weighted sum of a few sinusoids. Of course, the spectral content of natural sounds changes slowly over time. This is handled by time-frequency representations, such as the Short-Time Fourier Transform (STFT) and spectrogram, which indicate the spectral content of a local, windowed section of the sound. More specifically, the STFT ($x_{d,t}$) and spectrogram ($s_{d,t}$) of a discretised sound ($y_{t'}$) are given by,

$$x_{d,t} = \sum_{t'=1}^{T'} r_{t-t'} y_{t'} \exp\left(-i\omega_d t'\right), \quad s_{d,t} = \log |x_{d,t}|. \tag{1}$$

The (possibly frequency dependent) duration of the window ($r_{t-t'}$) must be chosen carefully, as it controls whether features are represented in the spectrum or in the time-variation of the spectra. For example, the window for speech is typically chosen to last for several pitch periods, so that both pitch and formant information is represented spectrally.

The first stage of the auditory pathway derives a time-frequency-like representation mechanically at the basilar membrane. Subsequent stages extract progressively more complex auditory features, with structure extending over more time. Thus, computational models of auditory processing often begin with a time-frequency (or auditory-filter bank) decomposition, deriving new representations from the time-frequency coefficients [4]. Machine-learning algorithms also typically operate on the time-frequency coefficients, and not directly on the waveform. The potential advantage lies in the ease with which auditory features may be extracted from the STFT representation. There are, however, associated problems. For example, time-frequency representations tend to be over-complete (e.g. the number of STFT coefficients tends to be larger than the number of samples of the original sound $T \times D > T'$). This means that realisable sounds live on a manifold in the time-frequency space (for the STFT this manifold is a hyper-plane). Algorithms that solve tasks like filling-in missing data or denoising must ensure that the new coefficients lie on the manifold. Typically this is achieved in an *ad hoc* manner by projecting time-frequency coefficients back onto the manifold according to an arbitrary metric [5]. For generative models of time-frequency coefficients, it is difficult to force the model to generate only on the realisable manifold. An alternative is to base a probabilistic model of the waveform on the same heuristics that led to the original time-frequency representation. Not only does this side-step the generation problem, but it also allows parameters of the representation, like the "window", to be chosen automatically.

The heuristic behind the STFT – that sound comprises sinusoids in slowly-varying linear superposition – led Qi et al [6] to propose a probabilistic algorithm called Bayesian Spectrum Estimation (BSE), in which the sinusoid coefficients ($x_{d,t}$) are latent variables. The forward model is,

$$p(x_{d,t}|x_{d,t-1}) = \text{Norm}\left(\lambda_d x_{d,t-1}, \sigma_d^2\right), \tag{2}$$

$$p(y_t|\mathbf{x}_t) = \text{Norm}\left(\sum_d x_{d,t} \sin\left(\omega_d t + \phi_d\right), \sigma_y^2\right). \tag{3}$$

The prior distribution over the coefficients is Gaussian and auto-regressive, evolving at a rate controlled by the dynamical parameters $\lambda_d$ and $\sigma_d^2$. Thus, as $\lambda_d \to 1$ and $\sigma_d^2 \to 0$ the processes become very slow, and as $\lambda_d \to 0$ and $\sigma_d^2 \to \infty$ they become very fast. More precisely, the length-scale of the coefficients is given by $\lambda_{\text{d}} = -\log(\lambda_d)$. The observations are generated by a weighted sum of sinusoids, plus Gaussian noise. This model is essentially a Linear Gaussian State Space System with time varying weights defined by the sinusoids. Thus, inference is simple, proceeding via the Kalman Smoother recursions with time-varying weights. In effect, these recursions dynamically adjust the window used to derive the coefficients, based on the past history of the stimulus. BSE is a model for the short-time structure of sounds and it will essentially form the bottom level of the MCP. In the next section we turn our attention to a model of the longer-time structure.

## 3 Probabilistic Demodulation Cascade

A salient property of the long-time statistics of sounds is the persistence of strong amplitude modulation [7]. Speech, for example, contains power in isolated regions corresponding to phonemes.

The phonemes themselves are localised into words, and then into sentences. Motivated by these observations, Anonymous Authors [8] have proposed a model for the long-time structures in sounds using a demodulation cascade. The basic idea of the demodulation cascade is to represent a sound as a product of processes drawn from a hierarchy, or cascade, of progressively longer time-scale modulators. For speech this might involve three processes: representing sentences on top, phonemes in the middle, and pitch and formants at the bottom (e.g. fig. 1A and B). To construct such a representation, one might start with a traditional amplitude demodulation algorithm, which decomposes a signal into a quickly-varying carrier and more slowly-varying envelope. The cascade could then be built by applying the same algorithm to the (possibly transformed) envelope, and then to the envelope that results from this, and so on. This procedure is only stable, however, if *both* the carrier *and* the envelope found by the demodulation algorithm are well-behaved. Unfortunately, traditional methods (like the Hilbert Transform, or low-pass filtering a non-linear transformation of the stimulus) return a suitable carrier *or* envelope, but not both. A new approach to amplitude demodulation is thus called for.

In a nutshell, the new approach is to view amplitude demodulation as a task of probabilistic inference. This is natural, as demodulation is fundamentally ill-posed — there are infinitely many decompositions of a signal into a positive envelope and real valued carrier — and so prior information must always be leveraged to realise such a decomposition. The generative model approach makes this information explicit. Furthermore, it not necessary to use the recursive procedure (just described) to derive a modulation cascade: the whole hierarchy can be estimated at once using a single generative model. The generative model for Probabilistic Amplitude Demodulation (PAD) is

$$p\left(z_0^{(m)}\right) = \text{Norm}\left(0,1\right), \quad p\left(z_t^{(m)}|z_{t-1}^{(m)}\right) = \text{Norm}\left(\lambda_m z_{t-1}^{(m)}, \sigma_m^2\right) \ \forall t > 0, \tag{4}$$

$$x_t^{(m)} = f_{a^{(m)}}\left(z_t^{(m)}\right) \ \forall m > 1, \qquad x_t^{(1)} = \text{Norm}\left(0,1\right), \qquad y_t = \prod_{m=1}^{M} x_t^{(m)}. \tag{5}$$

A set of modulators $(X_{2:M})$ are drawn in a two stage process: First a set of slowly varying processes $(Z_{2:M})$ are drawn from a one-step linear Gaussian prior (identical to Eq. 2). The effective length-scales of these processes, inherited by the modulators, are ordered such that $\lambda_m > \lambda_{m-1}$. Second the modulators are formed by passing these variables through a point-wise non-linearity to enforce positivity. A typical choice might be

$$f_{a^{(m)}}\left(z_t^{(m)}\right) \quad = \quad \log\left(\exp\left(z_t^{(m)} + a^{(m)}\right) + 1\right), \tag{6}$$

which is logarithmic for large negative values of $z_t^{(m)}$, and linear for large positive values. This transforms the Gaussian distribution over $z_t^{(m)}$ into a sparse, non-negative, distribution, which is a good match to the marginal distributions of natural envelopes. The parameter $a^{(m)}$ controls exactly where the transition from log to linear occurs, and consequently alters the degree of sparsity. These positive signals modulate a Gaussian white-noise carrier, to yield observations $y_{1:T}$ by a simple point-wise product. A typical draw from this generative model can be seen in Fig. 1C. This model is a fairly crude one for natural sounds. For example, as described in the previous section, we expect that the carrier process will be structured and yet it is modelled as Gaussian white noise. The surprising observation is that this very simple model is excellent at demodulation.

Inference in this model typically proceeds by a zero-temperature EM-like procedure. Firstly the carrier $(x_t^{(1)})$ is integrated out and then the modulators are found by *maximum a posteriori* (MAP). Slower, more Bayesian algorithms that integrate over the modulators using MCMC indicate that this approximation is not too severe, and the results are compelling.

## 4  Modulation Cascade Processes

We have reviewed two contrasting models: The first captures the local harmonic structure of sounds, but has no long-time structure; The second captures long-time amplitude modulations, but models the short-time structure as white noise. The goal of this section is to synthesise both to form a new model. We are guided by the observation that the auditory system might implement a similar synthesis. In the well-known psychophysical phenomenon of comodulation masking release (see [9] for a review), a tone masked by noise with a bandwidth greater than an auditory filter becomes audible

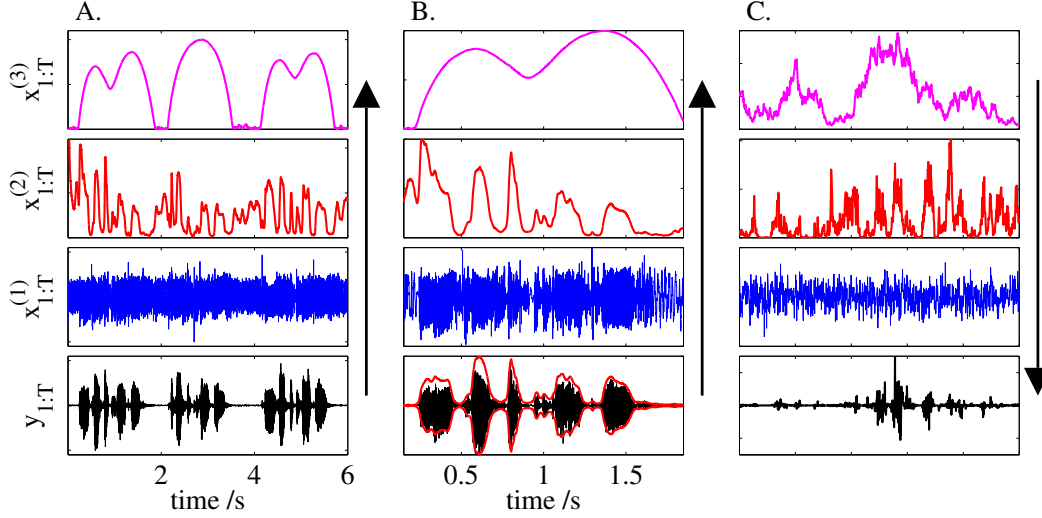

Figure 1: An example of a modulation-cascade representation of speech (A and B) and typical samples from generative models used to derive that representation (C). A) The spoken-speech waveform (black) is represented as the product of a carrier (blue), a phoneme modulator (red) and a sentence modulator (magenta). B) A close up of the first sentence (2 s) additionally showing the derived envelope $(x_t^{(2)} x_t^{(3)})$ superposed onto the speech (red, bottom panel). C) The generative model ($M = 3$) with a carrier (blue), a phoneme modulator (red) and a sentence modulator (magenta).

if the noise masker is amplitude modulated. This suggests that long-time envelope information is processed and analysed across (short-time) frequency channels in the auditory system.

A simple way to combine the two models would be to express each filter coefficient of the time-frequency model as a product of processes (e.g. $x_{d,t} = x_{d,t}^{(1)} x_{d,t}^{(2)}$). However, power across even widely seperated channels of natural sounds can be strongly correlated [7]. Furthermore, comodulation masking release suggests that amplitude-modulation is processed *across* frequency channels and not independently in each channel. Presumably this reflects the *collective* modulation of wide-band (or harmonic) sounds, with features that span many frequencies. Thus, a synthesis of BSE and PAD should incorporate the notion of auditory features.

**The forward model.**   The Modulation Cascade Process (MCP) is given by

$$p\left(z_{k_m,t}^{(m)}|z_{k_m,t-1}^{(m)},\theta\right) = \mathrm{Norm}\left(\lambda^{(m)}z_{k_m,t-1}^{(m)},\sigma_{(m)}^2\right) \quad m = 1:3, t > 0, \tag{7}$$

$$p\left(z_{k_m,0}^{(m)}\right) = \mathrm{Norm}\left(0,1\right), \quad x_{k_m,t}^{(m)} = f(z_{k_m,t}^{(m)},a^{(m)}) \quad m = 1:3, t \ge 0, \tag{8}$$

$$p\left(y_t|\mathbf{x}_t^{(m)},\theta\right) = \mathrm{Norm}\left(\mu_{y_t},\sigma_y^2\right), \quad \mu_{y_t} = \sum_{d,k_1,k_2} g_{d,k_1,k_2} x_{k_1,t}^{(1)} x_{k_2,t}^{(2)} x_t^{(3)} \sin\left(\omega_d t + \phi_d\right). \tag{9}$$

Once again, latent variables are arranged in a hierarchy according to their time-scales (which depend on $m$). At the top of the hierarchy is a long-time process which models slow structures, like the sentences of speech. The next level models more quickly varying structure (like phonemes). Finally, the bottom level of the hierarchy captures short-time variability (intra-phoneme variability for instance). Unlike in PAD, the middle and lower levels now contain multiple process. So, for example if $K_1 = 4$ and $K_2 = 2$, there would be four quickly varying modulators in the lower level, two modulators in the middle level, and one slowly varying modulator at the top (see fig. 2A).

The idea is that the modulators in the first level independently control the presence or absence of individual spectral features (given by $\sum_d g_{d,k_1,k_2} \sin\left(\omega_d t + \phi_d\right)$). For example, in speech a typical phoneme might be periodic, but this periodicity might change systematically as the speaker alters their pitch. This change in pitch might be modeled using two spectral features: one for the start of the phoneme and one for the end, with a region of coactivation in the middle. Indeed it is because speech

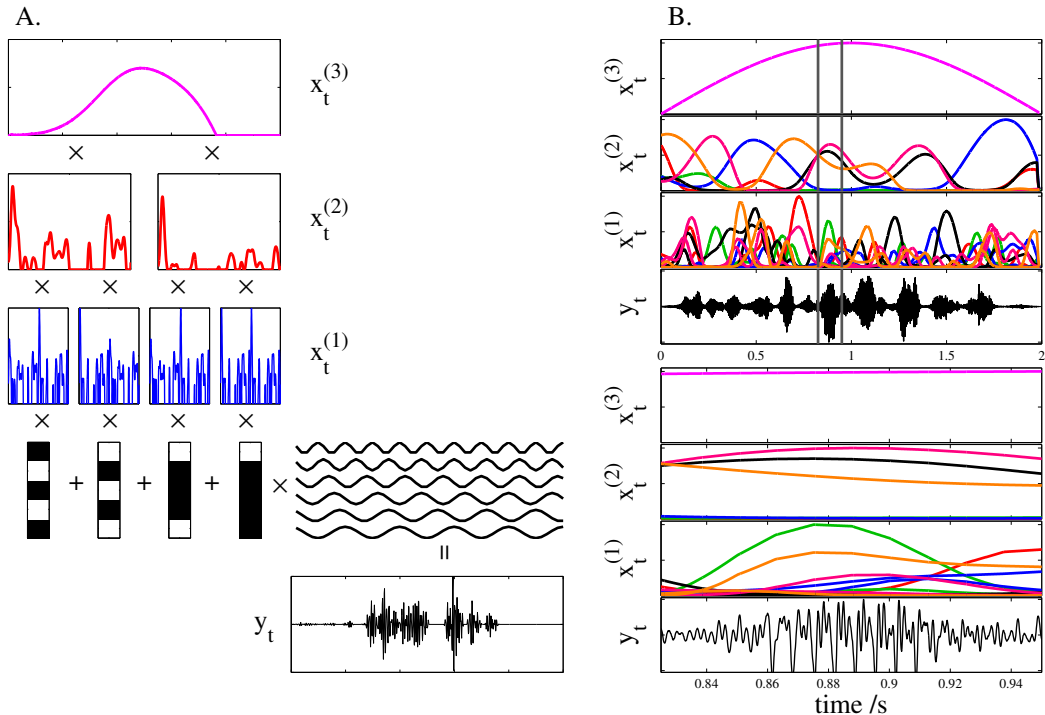

Figure 2: A. Schematic representation of the MCP forward model in the simple case when $K_1 = 4$, $K_2 = 2$ and $D = 6$. The hierarchy of latent variables moves from the slowest modulator at the top (magenta) to the fastest (blue) with an intermediate modulator between (red). The outer-product of the modulators multiplies the generative weights (black and white, only 4 of the 8 shown). In turn, these modulate sinusoids (top right) which are summed to produce the observations (bottom right). B. A draw from the forward model using parameters learned from a spoken-sentence (see the results section for more details of the model). The grey bars on the top four panels indicate the region depicted in the bottom four panels.

and other natural sounds are not precisely stationary even over short time-scales that we require the lowest layer of the hierarchy. The role of the modulators in the second level is to simultaneously turn on groups of similar features. For example, one modulator might control the presence of all the harmonic features and the other the broad-band features. Finally the top level modulator gates all the auditory features at once. Fig. 2B shows a draw from the forward model for a more complicated example. Promisingly the samples share many features of natural sounds.

**Relationship to other models.** This model has an interesting relationship to previous statistical models and in particular to the GSMMs. It is well known that when ICA is applied to data from natural scenes the inferred filter coefficients tend not to be independent (see [3, 10]), with coefficients corresponding to similar filters sharing power. GSMMs model dependencies using a hierarchical framework, in which the distribution over the coefficients depends on a set of latent variables that introduce correlations between their powers. The MCP is similar, in that the higher level latent variables alter the power of similar auditory features. Indeed, we suggest that the correlations in the power of ICA coefficients are a sign that AM is prevalent in natural scenes. The MCP can be seen as a generalisation of the GSMMs to include time-varying latent variables, a deeper hierarchy and a probabilistic time-frequency representation.

**Inference and learning algorithms.** Any type of learning in the MCP is computationally demanding. Motivated by speed, and encouraged by the results from PAD, the aim will therefore be to find a joint MAP estimate of the latent variables and the weights, that is

$$X, G \quad = \quad \underset{X,G}{\arg\max} \log p(X, Y, G|\theta). \tag{10}$$

Note that we have introduced a prior over the generative tensor. This prevents an undesirable feature of combined MAP and ML inference in such models: namely that the weights grow without bound, enabling the modal values of latent variables to shrink towards zero, increasing their density under the prior. The resulting cost function is,

$$
\begin{aligned}
\log p(\mathrm{X, Y, G}|\theta) \;=\; & \sum_{t=1}^{T} \log p(y_t|\mathbf{x}_t^{(1)}, \mathbf{x}_t^{(2)}) + \sum_{m=1}^{3} \sum_{k_m} \left( \sum_{t=1}^{T} \log p(\mathrm{z}_{k_m,t}^{(m)}|\mathrm{z}_{k_m,t-1}^{(m)}) \right. \\
& \left. + \sum_{t=0}^{T} \log \left| \frac{d\mathrm{z}_{k_m,t}^{(m)}}{d\mathrm{x}_{k_m,t}^{(m)}} \right| + \log p(\mathrm{z}_{k_m,0}^{(m)}) \right) + \sum_{k_1,k_2,d} \log p(\mathrm{g}_{d,k_1,k_2}) \quad (11)
\end{aligned}
$$

We would like to optimise this objective-function with respect to the latent variables ($\mathrm{x}_{k_m,t}^{(m)}$) and the generative tensor ($\mathrm{g}_{d,k_1,k_2}$). There are, however, two main obstacles. The first is that there are a large number of latent variables to estimate ($T \times (K_1 + K_2)$), making inference slow. The second is that the generative tensor contains a large number of elements $D \times K_1 \times K_2$, making learning slow too. The solution is to find a good initialisation procedure, and then to fine-tune using a slow EM-like algorithm that iterates between updating the latents and the weights. First we outline the initialisation procedure.

The key to learning complicated hierarchical models is to initialise well, and so the procedure developed for the MCP will be explained in some detail. The main idea is to learn the model one layer at a time. This is achieved by clamping the upper layers of the hierarchy that are not being learned to unity. In the first stage of the initialisation, for example, the top and middle levels of the hierarchy are clamped and the mean of the emission distribution becomes

$$
\mu_{\mathrm{y}_t} \;=\; \sum_{d,k_1} \gamma_{d,k_1} \mathrm{x}_{k_1,t}^{(1)} \sin\left(\omega_d t + \phi_d\right), \tag{12}
$$

where $\gamma_{d,k_1} = \sum_{k_2} \mathrm{g}_{d,k_1,k_2}$. Learning and inference then proceed by gradient based optimisation of the cost-function ($\log p(\mathrm{X, Y, G}|\theta)$) with respect to the un-clamped latents ($\mathrm{x}_{k_1,t}^{(1)}$) and the contracted generative weights ($\gamma_{d,k1}$). This is much faster than the full optimisation as there are both fewer latents and fewer parameters to estimate. When this process is complete, the second layer of latent variables is un-clamped, and learning of these variables commences. This requires the full generative tensor, which must be initialised from the contracted generative weights learned at the previous stage. One choice is to set the individual weights to their averages $\mathrm{g}_{d,k_1,k_2} = \frac{1}{K_2}\gamma_{d,k_1}$ and this works well, but empirically slows learning. An alternative is to use small chunks of sounds to learn the lower level weights. These chunks are chosen to be relatively stationary segments that have a time-scale similar to the second-level modulators. This allows us to make the simplifying assumption that just one second-level modulator was active during the chunk. The generative tensor can be therefore be initialised using $\mathrm{g}_{d,k_1,k_2} = \gamma_{d,k_1}\delta_{k_2,j}$. Typically this method causes the second stage of learning to converge faster, and to a similar solution.

In contrast to the initialisation, the fine tuning algorithm is simple. In the E-Step the latent variables are updated simultaneously using gradient based optimisation of Eq. 11. In the M-Step, the generative tensor is updated using co-ordinate ascent. That is to say that we sequentially update each $\mathbf{g}_{k_1,k_2}$ using gradient based optimisation of Eq. 11 and iterate over $k_1$ and $k_2$. In principle, joint optimisation of the generative tensor and latent variables is possible, but the memory requirements are prohibitive. This is also why co-ordinate ascent is used to learn the generative tensor (rather than using the usual linear regression solution which involves a prohibitive matrix inverse).

## 5  Results

The MCP was trained on a spoken sentence, lasting 2s and sampled at 8000Hz, using the algorithm outlined in the previous section. The time-scales of the modulators were chosen to be $\{20 \text{ ms}, 200 \text{ ms}, 2 \text{ s}\}$. The time-frequency representation had $D/2 = 100$ sines and $D/2 = 100$ cosines spaced logarithmically from $100 - 4000$Hz. The model was given $K_1 = 18$ latent variables in the first level of the hierarchy, and $K_2 = 6$ in the second. Learning took 16hrs to run on a 2.2 GHz Opteron with 2Gb of memory.

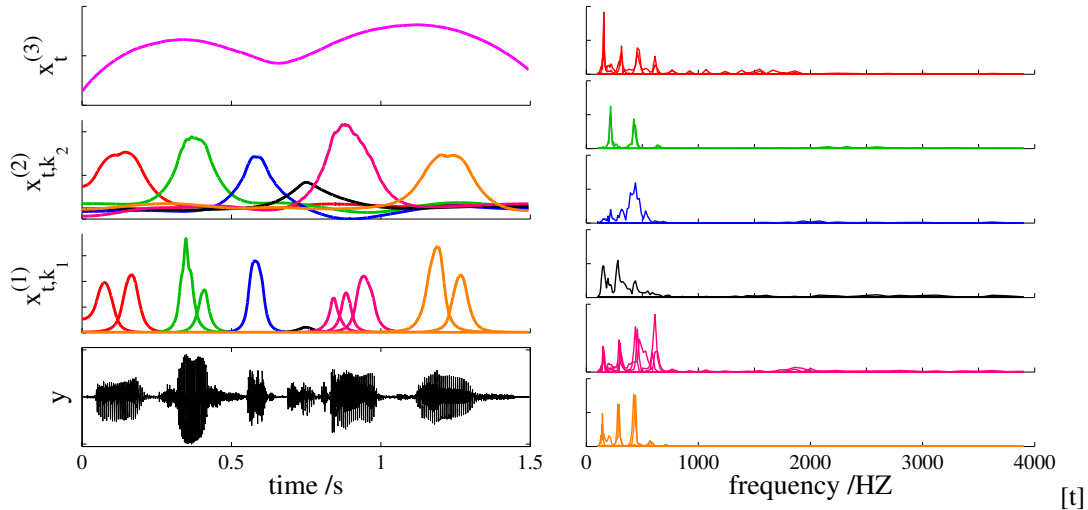

Figure 3: Application of the MCP to speech. Left panels: The inferred latent variable hierarchy. At top is the sentence modulator (magenta). Next are the phoneme modulators, followed by the intra-phoneme modulators. These are coloured according to which of the phoneme modulators they interact most strongly with. The speech waveform is shown in the bottom panel. Right panels: The learned spectral features ($\sqrt{g_{sin}^2 + g_{cos}^2}$) coloured according to phoneme modulator. For example, the top panel show the spectra from $\mathbf{g}_{k_1=1:18,k_2=1}$. Spectra corresponding to one phoneme modulator look similar and offer the features only differ in their phase.

The results can be seen in Fig. 3. The MCP recovers a sentence modulator, phoneme modulators, and intra-phoneme modulators. Typically a pair of features are used to model a phoneme, and often they have similar spectra as expected. The spectra fall into distinct classes: those which are harmonic (modelling voiced features) and those which are broad-band (modelling unvoiced features). One way of assessing which features of speech the model captures is to sample from the forward model using the learned parameters. This can be seen in Fig. 2B. The conclusion is that the model is capturing structure across a wide range of time-scales: formants and pitch structure, phoneme structure, and sentence structure.

There are, however, two noticeable differences between the real and generated data. The first is that the generated data contain fewer transients and noise segments than natural speech, and more vowel-like components. The reason for this is that at the sampling rates used, many of the noisy segments are indistinguishable from white-noise and are explained using observation noise. These problems are alleviated by moving to higher sampling rates, but the algorithm is then markedly slower. The second difference concerns the inferred and generated latent variables in that the former are much sparser than the latter. The reason is that learned generative tensor contains many $\mathbf{g}_{k_1,k_2}$ which are nearly zero. In generation, this means that significant contributions to the output are only made when particular pairs of phoneme and intra-phoneme modulators are active. So although many modulators are active at one time, only one or two make sizeable contributions. Conversely, in inference, we can only get information about the value of a modulator when it is part of a contributing pair. If this is not the case, the inference goes to the maximum of the prior which is zero. In effect there are large error-bars on the non-contributing components' estimates.

Finally, to indicate the improvement of the MCP over PAD and BSE, we compare the algorithms abilities to fill in missing sections of a spoken sentence. The average root-mean-squared (RMS) error per sample is used as a metric to compare the algorithms. In order to use the MCP to fill in the missing data, it is first necessary to learn a set of auditory features. The MCP was therefore trained on a different spoken sentence from the same speaker, before inference was carried out on the test data. To make the comparison fair, BSE is given an identical set of sinusoidal basis functions as MCP, and the associated smoothness priors were learned on the same training data.

Typical results can be seen in fig. 4. On average the RMS errors for MCP, BSE and PAD were: $\{0.10, 0.30, 0.41\}$. As PAD models the carrier as white noise it predicts zeros in the missing regions

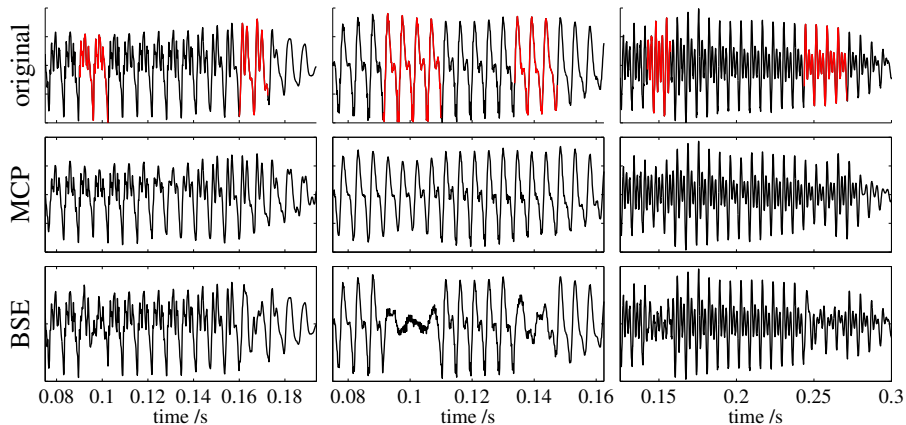

Figure 4: A selection of typical missing data results for three phonemes (columns). The top row shows the original speech segement with the missing regions shown in red. The middle row shows the predictions made by the MCP and the bottom row those made by BSE.

and therefore it merely serves as a baseline in these experiments. Both MCP and BSE smoothly interpolate their latent variables over the missing region. However, whereas BSE smoothly interpolates each sinusoidal component independently, MCP interpolates the set of learned auditory features in a complex manner determined by the interaction of the modulators. It is for this reason that it improves over BSE by such a large margin.

## 6 Conclusion

We have introduced a neuroscience-inspired generative model for natural sounds that is capable of capturing structure spanning a wide range of temporal scales. The model is a marriage between a probabilistic time-frequency representation (that captures the short-time structure) and a probabilistic demodulation cascade (that captures the long-time structure). When the model is trained on a spoken sentence, the first level of the hierarchy learns auditory features (weighted sets of sinusoids) that capture structures like different voiced sections of speech. The upper levels comprise a temporally ordered set of modulators are used to represent sentence structure, phoneme structure and intra-phoneme variability. The superiority of the new model over its parents was demonstrated in a missing data experiment where it out-performed the Bayesian time-frequency analysis by a large margin.

## References

[1] Bregman, A.S. (1990) *Auditory Scene Analysis.* MIT Press.

[2] Smith E. & Lewicki, M.S. (2006) Efficient Auditory Coding. *Nature* 439 (7079).

[3] Simoncelli, E.P. (2003) Vision and the statistics of the visual environment. *Curr Opin Neurobi* 13(2):144-9.

[4] Patterson, R.D. (2000) Auditory images: How complex sounds are represented in the auditory system. *J Acoust Soc Japan (E)* 21(4):183-190.

[5] Griffin, D. & Lim J. (1984) Signal estimation from modified short-time Fourier transform. *IEEE Trans. on ASSP* 32(2):236-243.

[6] Qi, Y., Minka, T. & Picard, R.W. (2002) Bayesian Spectrum Estimation of Unevenly Sampled Nonstationary Data. MIT Media Lab Technical Report Vismod-TR-556.

[7] Attias, H. & Schreiner, C.E. (1997) Low-Order Temporal Statistics of Natural Sounds. *Adv in Neural Info Processing Sys 9*. MIT Press.

[8] Anonymous Authors (2007) Probabilistic Amplitude Demodulation. ICA 2007 Conference Proceedings. Springer, in press.

[9] Moore, B.C.J. (2003) *An Introduction to the Psychology of Hearing.* Academic Press.

[10] Karklin, Y. & Lewicki, M.S. (2005) A hierarchical Bayesian model for learning nonlinear statistical regularities in nonstationary natural signals. *Neural Comput* 17(2):397-423.

